# Oscillatory Neural Fields for Globally Optimal Path Planning

**Michael Lemmon**
Dept. of Electrical Engineering
University of Notre Dame
Notre Dame, Indiana 46556

## Abstract

A neural network solution is proposed for solving path planning problems faced by mobile robots. The proposed network is a two-dimensional sheet of neurons forming a distributed representation of the robot's workspace. Lateral interconnections between neurons are "cooperative", so that the network exhibits oscillatory behaviour. These oscillations are used to generate solutions of Bellman's dynamic programming equation in the context of path planning. Simulation experiments imply that these networks locate global optimal paths even in the presence of substantial levels of circuit noise.

## 1 Dynamic Programming and Path Planning

Consider a 2-DOF robot moving about in a 2-dimensional world. A robot's location is denoted by the real vector, $\mathbf{p}$. The collection of all locations forms a set called the workspace. An admissible point in the workspace is any location which the robot may occupy. The set of all admissible points is called the free workspace. The free workspace's complement represents a collection of obstacles. The robot moves through the workspace along a path which is denoted by the parameterized curve, $\mathbf{p}(t)$. An admissible path is one which lies wholly in the robot's free workspace. Assume that there is an initial robot position, $\mathbf{p}_0$, and a desired final position, $\mathbf{p}_f$. The robot path planning problem is to find an admissible path with $\mathbf{p}_0$ and $\mathbf{p}_f$ as endpoints such that some "optimality" criterion is satisfied.

The path planning problem may be stated more precisely from an optimal control theorist's viewpoint. Treat the robot as a dynamic system which is characterized by a state vector, p, and a control vector, u. For the highest levels in a control hierarchy, it can be assumed that the robot's dynamics are modeled by the differential equation, $\dot{p} = u$. This equation says that the state velocity equals the applied control. To define what is meant by "optimal", a performance functional is introduced.

$$J(\mathbf{u}) = ||\mathbf{p}(t_f) - \mathbf{p}_f||^2 + \int_0^{t_f} c(\mathbf{p})\mathbf{u}^t\mathbf{u}\,dt \tag{1}$$

where $||\mathbf{x}||$ is the norm of vector $\mathbf{x}$ and where the functional $c(\mathbf{p})$ is unity if $\mathbf{p}$ lies in the free workspace and is infinite otherwise. This weighting functional is used to insure that the control does not take the robot into obstacles. Equation 1's optimality criterion minimizes the robot's control effort while penalizing controls which do not satisfy the terminal constraints.

With the preceding definitions, the optimal path planning problem states that for some final time, $t_f$, find the control $\mathbf{u}(t)$ which minimizes the performance functional $J(\mathbf{u})$. One very powerful method for tackling this minimization problem is to use dynamic programming (Bryson, 1975). According to dynamic programming, the optimal control, $\mathbf{u}_{opt}$, is obtained from the gradient of an "optimal return function", $J^o(\mathbf{p})$. In other words, $\mathbf{u}_{opt} = \nabla J^o$. The optimal return functional satisfies the Hamilton-Jacobi-Bellman (HJB) equation. For the dynamic optimization problem given above, the HJB equation is easily shown to be

$$\frac{\partial J^o}{\partial t} = \begin{cases} -\frac{1}{4}(\nabla J^o)^t(\nabla J^o) & c(\mathbf{p}) = 1 \\ 0 & c(\mathbf{p}) = \infty \end{cases} \tag{2}$$

This is a first order nonlinear partial differential equation (PDE) with terminal (boundary) condition, $J^o(t_f) = ||\mathbf{p}(t_f) - \mathbf{p}_f||^2$. Once equation 2 has been solved for the $J^o$, then the optimal "path" is determined by following the gradient of $J^o$.

Solutions to equation 2 must generally be obtained numerically. One solution approach numerically integrates a full discretization of equation 2 backwards in time using the terminal condition, $J^o(t_f)$, as the starting point. The proposed numerical solution is attempting to find characteristic trajectories of the nonlinear first-order PDE. The PDE nonlinearities, however, only insure that these characteristics exist locally (i.e., in an open neighborhood about the terminal condition). The resulting numerical solutions are, therefore, only valid in a "local" sense. This is reflected in the fact that truncation errors introduced by the discretization process will eventually result in numerical solutions violating the underlying principle of optimality embodied by the HJB equation.

In solving path planning problems, local solutions based on the numerical integration of equation 2 are not acceptable due to the "local" nature of the resulting solutions. Global solutions are required and these may be obtained by solving an associated variational problem (Benton, 1977). Assume that the optimal return function at time $t_f$ is known on a closed set $B$. The variational solution for equation 2 states that the optimal return at time $t < t_f$ at a point $\mathbf{p}$ in the neighborhood of the boundary set $B$ will be given by

$$J^o(\mathbf{p}, t) = \min_{\mathbf{y} \in B} \left\{ J^o(\mathbf{y}, t_f) + \frac{||\mathbf{p} - \mathbf{y}||^2}{(t_f - t)} \right\} \tag{3}$$

where $\|\mathbf{p}\|$ denotes the $L_2$ norm of vector p. Equation 3 is easily generalized to other vector norms and only applies in regions where $c(\mathbf{p}) = 1$ (i.e. the robot's free workspace). For obstacles, $J^o(\mathbf{p}, t) = J^o(\mathbf{p}, t_f)$ for all $t < t_f$. In other words, the optimal return is unchanged in obstacles.

## 2   Oscillatory Neural Fields

The proposed neural network consists of $MN$ neurons arranged as a 2-d sheet called a "neural field". The neurons are put in a one-to-one correspondence with the ordered pairs, $(i, j)$ where $i = 1, \ldots, N$ and $j = 1, \ldots, M$. The ordered pair $(i, j)$ will sometimes be called the $(i, j)$th neuron's "label". Associated with the $(i, j)$th neuron is a set of neuron labels denoted by $\mathbf{N}_{i,j}$. The neurons' whose labels lie in $\mathbf{N}_{i,j}$ are called the "neighbors" of the $(i, j)$th neuron.

The $(i, j)$th neuron is characterized by two states. The short term activity (STA) state, $x_{i,j}$, is a scalar representing the neuron's activity in response to the currently applied stimulus. The long term activity (LTA) state, $w_{i,j}$, is a scalar representing the neuron's "average" activity in response to recently applied stimuli. Each neuron produces an output, $f(x_{i,j})$, which is a unit step function of the STA state. (i.e., $f(x) = 1$ if $x > 0$ and $f(x) = 0$ if $x \leq 0$). A neuron will be called "active" or "inactive" if its output is unity or zero, respectively.

Each neuron is also characterized by a set of constants. These constants are either externally applied inputs or internal parameters. They are the disturbance $y_{i,j}$, the rate constant $\lambda_{i,j}$, and the position vector $\mathbf{p}_{i,j}$. The position vector is a 2-d vector mapping the neuron onto the robot's workspace. The rate constant models the STA state's underlying dynamic time constant. The rate constant is used to encode whether or not a neuron maps onto an obstacle in the robot's workspace. The external disturbance is used to initiate the network's search for the optimal path.

The evolution of the STA and LTA states is controlled by the state equations. These equations are assumed to change in a synchronous fashion. The STA state equation is

$$x_{i,j}^+ = G\left( x_{i,j}^- + \lambda_{i,j} y_{i,j} + \lambda_{i,j} \sum_{(k,l) \in \mathbf{N}_{i,j}} D_{kl} f(x_{k,l}) \right) \tag{4}$$

where the summation is over all neurons contained within the neighborhood, $\mathbf{N}_{i,j}$, of the $(i, j)$th neuron. The function $G(x)$ is zero if $x < 0$ and is $x$ if $x \geq 0$. This function is used to prevent the neuron's activity level from falling below zero. $D_{kl}$ are network parameters controlling the strength of lateral interactions between neurons. The LTA state equation is

$$w_{i,j}^+ = w_{i,j}^- + |f'(x_{i,j})| \tag{5}$$

Equation 5 means that the LTA state is incremented by one every time the $(i, j)$th neuron's output changes.

Specific choices for the interconnection weights result in oscillatory behaviour. The specific network under consideration is a cooperative field where $D_{kl} = 1$ if $(k, l) \neq$

$(i, j)$ and $D_{kl} = -A < 0$ if $(k, l) = (i, j)$. Without loss of generality it will also be assumed that the external disturbances are bounded between zero and one. It is also assumed that the rate constants, $\lambda_{i,j}$ are either zero or unity. In the path planning application, rate constants will be used to encode whether or not a given neuron represents an obstacle or a point in the free-workspace. Consequently, any neuron where $\lambda_{i,j} = 0$ will be called an "obstacle" neuron and any neuron where $\lambda_{i,j} = 1$ will be called a "free-space" neuron. Under these assumptions, it has been shown (Lemmon, 1991a) that once a free-space neuron turns active it will be oscillating with a period of 2 provided it has at least one free-space neuron as a neighbor.

## 3    Path Planning and Neural Fields

The oscillatory neural field introduced above can be used to generate solutions of the Bellman iteration (Eq. 3) with respect to the supremum norm. Assume that all neuron STA and LTA states are zero at time 0. Assume that the position vectors form a regular grid of points, $p_{i,j} = (i\Delta, j\Delta)^t$ where $\Delta$ is a constant controlling the grid's size. Assume that all external disturbances but one are zero. In other words, for a specific neuron with label $(i, j)$, $y_{k,l} = 1$ if $(k, l) = (i, j)$ and is zero otherwise. Also assume a neighborhood structure where $N_{i,j}$ consist of the $(i, j)$th neuron and its eight nearest neighbors, $N_{i,j} = \{(i + k, j + l); k = -1, 0, 1; l = -1, 0, 1\}$. WIth these assumptions it has been shown (Lemmon, 1991a) that the LTA state for the $(i, j)$th neuron at time $n$ will be given by $G(n - \rho_{kl})$ where $\rho_{kl}$ is the length of the shortest path from $p_{k,l}$ and $p_{i,j}$ with respect to the supremum norm.

This fact can be seen quite clearly by examining the LTA state's dynamics in a small closed neighborhood about the $(k, l)$th neuron. First note that the LTA state equation simply increments the LTA state by one every time the neuron's STA state toggles its output. Since a neuron oscillates after it has been initially activated, the LTA state, will represent the time at which the neuron was first activated. This time, in turn, will simply be the "length" of the shortest path from the site of the initial distrubance. In particular, consider the neighborhood set for the $(k, l)$th neuron and let's assume that the $(k, l)$th neuron has not yet been activated. If the neighbor has been activated, with an LTA state of a given value, then we see that the $(k, l)$th neuron will be activated on the next cycle and we have

$$w_{k,l} = \max_{(m,n) \in N_{k,l}} \left( w_{m,n} - \frac{\|p_{k,l} - p_{m,n}\|_\infty}{\Delta} \right) \qquad (6)$$

This is simply a dual form of the Bellman iteration shown in equation 3. In other words, over the free-space neurons, we can conclude that the network is solving the Bellman equation with respect to the supremum norm.

In light of the preceding discussion, the use of cooperative neural fields for path planning is straightforward. First apply a disturbance at the neuron mapping onto the desired terminal position, $p_f$ and allow the field to generate STA oscillations. When the neuron mapping onto the robot's current position is activated, stop the oscillatory behaviour. The resulting LTA state distribution for the $(i, j)$th neuron equals the negative of the minimum distance (with respect to the sup norm) from that neuron to the initial disturbance. The optimal path is then generated by a sequence of controls which ascends the gradient of the LTA state distribution.

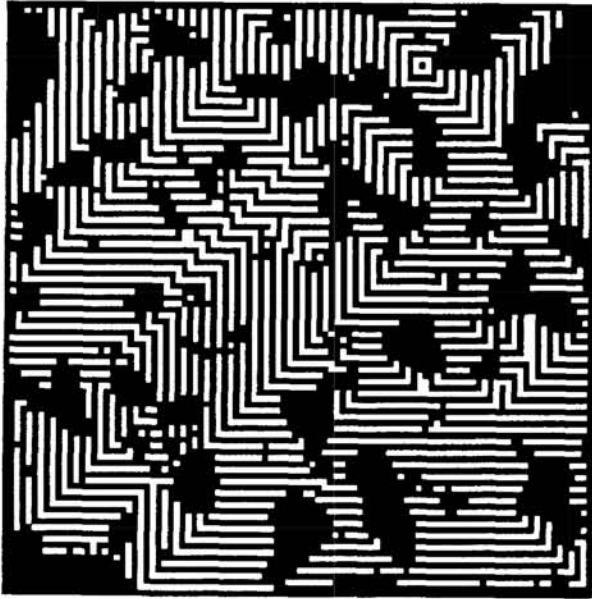

fig 1. STA activity waves

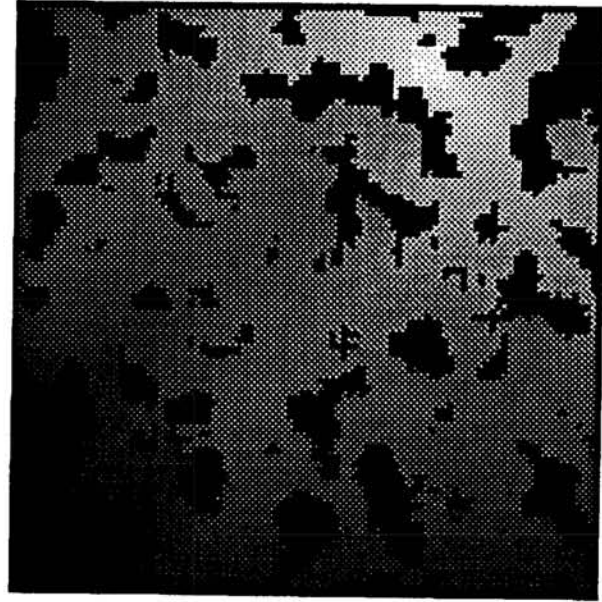

fig 2. LTA distribution

Several simulations of the cooperative neural path planner have been implemented. The most complex case studied by these simulations assumed an array of 100 by 100 neurons. Several obstacles of irregular shape and size were randomly distributed over the workspace. An initial disturbance was introduced at the desired terminal location and STA oscillations were observed. A snapshot of the neuronal outputs is shown in figure 1. This figure clearly shows wavefronts of neuronal activity propagating away from the initial disturbance (neuron (70,10) in the upper right hand corner of figure 1). The "activity" waves propagate around obstacles without any reflections. When the activity waves reach the neuron mapping onto the robot's current position, the STA oscillations were turned off. The LTA distribution resulting from this particular simulation run is shown in figure 2. In this figure, light regions denote areas of large LTA state and dark regions denote areas of small LTA state.

The generation of the optimal path can be computed as the robot is moving towards its goal. Let the robot's current position be the $(i, j)$th neuron's position vector. The robot will then generate a control which takes it to the position associated with one of the $(i, j)$th neuron's neighbors. In particular, the control is chosen so that the robot moves to the neuron whose LTA state is largest in the neighborhood set, $N_{i,j}$. In other words, the next position vector to be chosen is $p_{k,l}$ such that its LTA state is

$$w_{k,l} = \max_{(x,y) \in N_{i,j}} w_{x,y} \qquad (7)$$

Because of the LTA distribution's optimality property, this local control strategy is guaranteed to generate the optimal path (with respect to the sup norm) connecting the robot to its desired terminal position. It should be noted that the selection of the control can also be done with an analog neural network. In this case, the LTA

states of neurons in the neighborhood set, $\mathbf{N}_{i,j}$ are used as inputs to a competitively inhibited neural net. The competitive interactions in this network will always select the direction with the largest LTA state.

Since neuronal dynamics are analog in nature, it is important to consider the impact of noise on the implementation. Analog systems will generally exhibit noise levels with effective dynamic ranges being at most 6 to 8 bits. Noise can enter the network in several ways. The LTA state equation can have a noise term (LTA noise), so that the LTA distribution may deviate from the optimal distribution. In our experiments, we assumed that LTA noise was additive and white. Noise may also enter in the selection of the robot's controls (selection noise). In this case, the robot's next position is the position vector, $\mathbf{p}_{k,l}$ such that $w_{k,l} = \max_{(x,y)\in \mathbf{N}_{i,j}}(w_{x,y} + v_{x,y})$ where $v_{x,y}$ is an i.i.d array of stochastic processes. Simulation results reported below assume that the noise processes, $v_{x,y}$, are positive and uniformly distributed i.i.d. processes. The introduction of noise places constraints on the "quality" of individual neurons, where quality is measured by the neuron's effective dynamic range. Two sets of simulation experiments have been conducted to assess the neural field's dynamic range requirements. In the following simulations, dynamic range is defined by the equation $-\log_2|v_m|$, where $|v_m|$ is the maximum value the noise process can take. The unit for this measure of dynamic range is "bits".

The first set of simulation experiments selected robotic controls in a noisy fashion. Figure 3 shows the paths generated by a simulation run where the signal to noise ratio was 1 (0 bits). The results indicate that the impact of "selection" noise is to "confuse" the robot so it takes longer to find the desired terminal point. The path shown in figures 3 represents a random walk about the true optimal path. The important thing to note about this example is that the system is capable of tolerating extremely large amounts of "selection" noise.

The second set of simulation experiments introduced LTA noise. These noise experiments had a detrimental effect on the robot's path planning abilities in that several spurious extremals were generated in the LTA distribution. The result of the spurious extremals is to fool the robot into thinking it has reached its terminal destination when in fact it has not. As noise levels increase, the number of spurious states increase. Figure 4, shows how this increase varies with the neuron's effective dynamic range. The surprising thing about this result is that for neurons with as little as 3 bits of effective dynamic range the LTA distribution is free of spurious maxima. Even with less than 3 bits of dynamic range, the performance degradation is not catastrophic. LTA noise may cause the robot to stop early; but upon stopping the robot is closer to the desired terminal state. Therefore, the path planning module can be easily run again and because the robot is closer to its goal there will be a greater probability of success in the second trial.

## 4    Extensions and Conclusions

This paper reported on the use of oscillatory neural networks to solve path planning problems. It was shown that the proposed neural field can compute dynamic programming solutions to path planning problems with respect to the supremeum norm. Simulation experiments showed that this approach exhibited low sensitivity

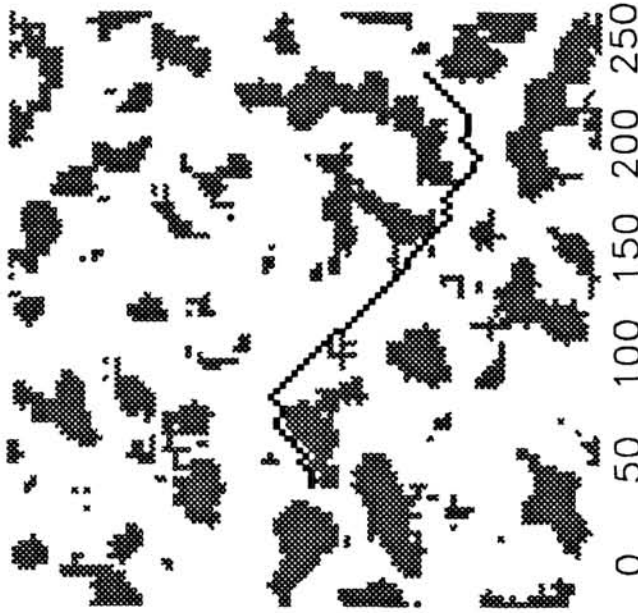

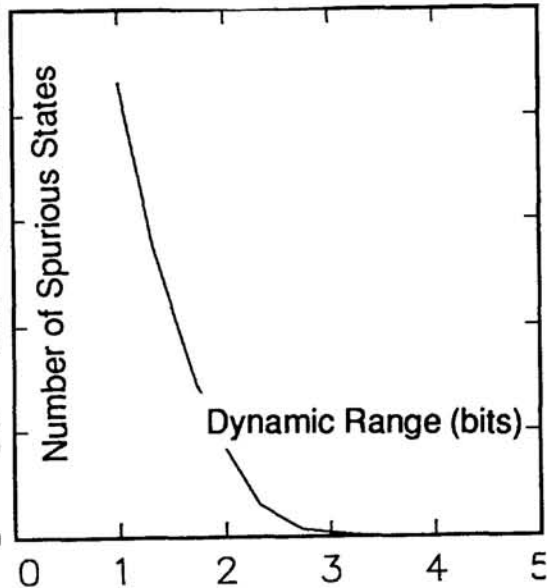

fig 3. Selected Path                                    fig 4. Dynamic Range

to noise, thereby supporting the feasibility of analog VLSI implementations.

The work reported here is related to resistive grid approaches for solving optimization problems (Chua, 1984). Resistive grid approaches may be viewed as "passive" relaxation methods, while the oscillatory neural field is an "active" approach. The primary virtue of the "active" approach lies in the network's potential to control the optimization criterion by selecting the interconnections and rate constants. In this paper and (Lemmon, 1991a), lateral interconnections were chosen to induce STA state oscillations and this choice yields a network which solves the Bellman equation with respect to the supremum norm. A slight modification of this model is currently under investigation in which the neuron's dynamics directly realize the iteration of equation 6 with respect to more general path metrics. This analog network is based on an SIMD approach originally proposed in (Lemmon, 1991). Results for this field are shown in figures 5 and 6. These figures show paths determined by networks utilizing different path metrics. In figure 5, the network penalizes movement in all directions equally. In figure 6, there is a strong penalty for horizontal or vertical movements. As a result of these penalties (which are implemented directly in the interconnection constants $D_{kl}$), the two networks' "optimal" paths are different. The path in figure 6 shows a clear preference for making diagonal rather than vertical or horizontal moves. These results clearly demonstrate the ability of an "active" neural field to solve path planning problems with respect to general path metrics. These different path metrics, of course, represent constraints on the system's path planning capabilities and as a result suggest that "active" networks may provide a systematic way of incorporating holonomic and nonholonomic constraints into the path planning process.

A final comment must be made on the apparent complexity of this approach.

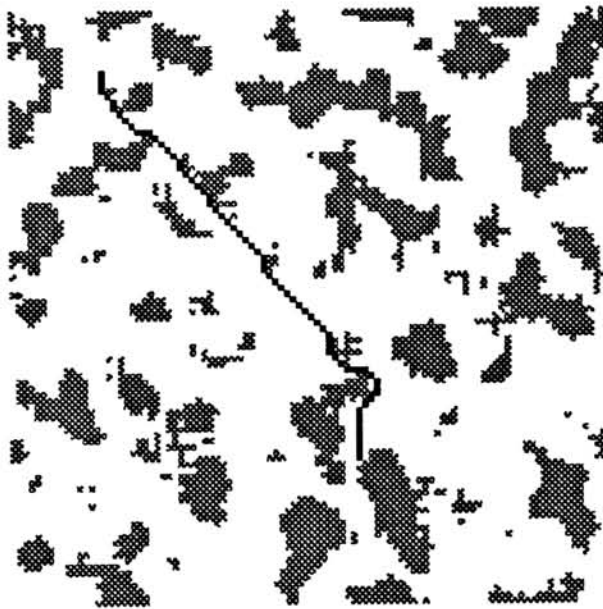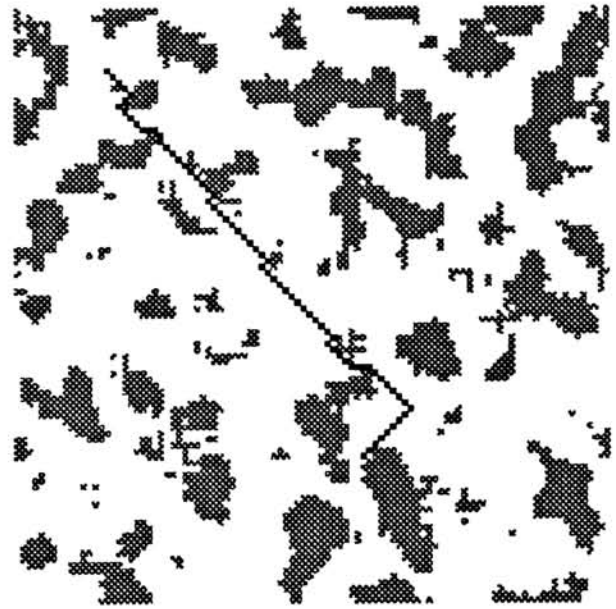

fig 5. No Direction Favored                fig 6. Diagonal Direction Favored

Clearly, if this method is to be of practical significance, it must be extended beyond the 2-DOF problem to arbitrary task domains. This extension, however, is nontrivial due to the "curse of dimensionality" experienced by straightforward applications of dynamic programming. An important area of future research therefore addresses the decomposition of real-world tasks into smaller subtasks which are amenable to the solution methodology proposed in this paper.

## Acknowledgements

I would like to acknowledge the partial financial support of the National Science Foundation, grant number NSF-IRI-91-09298.

## References

S.H. Benton Jr., (1977) *The Hamilton-Jacobi equation: A Global Approach.* Academic Press.

A.E. Bryson and Y.C. Ho, (1975) *Applied Optimal Control*, Hemisphere Publishing. Washington D.C.

L.O. Chua and G.N. Lin, (1984) Nonlinear programming without computation, *IEEE Trans. Circuits Syst.*, **CAS-31**:182-188

M.D. Lemmon, (1991) Real time optimal path planning using a distributed computing paradigm, *Proceedings of the Americal Control Conference*, Boston, MA, June 1991.

M.D. Lemmon, (1991a) 2-Degree-of-Freedom Robot Path Planning using Cooperative Neural Fields. *Neural Computation* 3(3):350-362.